# Logistic Regression for Single Trial EEG Classification

**Ryota Tomioka**[*]      **Kazuyuki Aihara**[†]
Dept. of Mathematical Informatics,
IST, The University of Tokyo,
113-8656 Tokyo, Japan.
`ryotat@first.fhg.de`
`aihara@sat.t.u-tokyo.ac.jp`

**Klaus-Robert Müller**[*]
Dept. of Computer Science,
Technical University of Berlin,
Franklinstr. 28/29,
10587 Berlin, Germany.
`klaus@first.fhg.de`

## Abstract

We propose a novel framework for the classification of single trial ElectroEncephaloGraphy (EEG), based on regularized logistic regression. Framed in this robust statistical framework no prior feature extraction or outlier removal is required. We present two variations of parameterizing the regression function: (a) with a full rank symmetric matrix coefficient and (b) as a difference of two rank=1 matrices. In the first case, the problem is convex and the logistic regression is optimal under a generative model. The latter case is shown to be related to the Common Spatial Pattern (CSP) algorithm, which is a popular technique in Brain Computer Interfacing. The regression coefficients can also be topographically mapped onto the scalp similarly to CSP projections, which allows neuro-physiological interpretation. Simulations on 162 BCI datasets demonstrate that classification accuracy and robustness compares favorably against conventional CSP based classifiers.

## 1 Introduction

The goal of Brain-Computer Interface (BCI) research [1, 2, 3, 4, 5, 6, 7] is to provide a direct control pathway from human intentions reflected in brain signals to computers. Such a system will not only provide disabled people more direct and natural control over a neuro-prosthesis or over a computer application (e.g. [2]) but also opens up a further channel of man machine interaction for healthy people to communicate solely by their intentions.

Machine learning approaches to BCI have proven to be effective by requiring less subject training and by compensating for the high inter-subject variability. In this field, a number of studies have focused on constructing better low dimensional representations that combine various features of brain activities [3, 4], because the problem of classifying EEG signals is intrinsically high dimensional. In particular, efforts have been made to reduce the number of electrodes by eliminating electrodes recursively [8] or by decomposition techniques e.g., ICA, which only uses the marginal distribution, or Common Spatial Patterns (CSP) [9] which additionally takes the labels into account. In practice, often a BCI system has been constructed by combining a feature extraction step and a classification step.

Our contribution is a logistic regression classifier that integrates both steps under the roof of a single minimization problem and uses well controlled regularization. Moreover, the classifier output has a probabilistic interpretation. We study a BCI based on the motor

---
[*]Fraunhofer FIRST.IDA, Kekuléstr. 7, 12489 Berlin, Germany.
[†]ERATO Aihara Complexity Modeling Project, JST, 153-8505 Tokyo, Japan

imagination paradigm. Motor imagination can be captured through spatially localized band-power modulation in the $\mu$- (10-15Hz) or $\beta$- (20-30Hz) band characterized by the second-order statistics of the signal; the underlying neuro-physiology is well known as Event Related Desynchronization (ERD) [10].

## 1.1  Problem setting

Let us denote by $X \in \mathbb{R}^{d \times T}$ the EEG signal of a single trial of an imaginary motor move-ment[1], where $d$ is the number of electrodes and $T$ is the number of sampled time-points in a trial. We consider a binary classification problem where each class, e.g. right or left hand imaginary movement, is called positive (+) or negative (−) class. Let $y \in \{+1, -1\}$ be the class label. Given a set of trials and labels $\{X_i, y_i\}_{i=1}^n$, the task is to predict the class label $y$ for an unobserved trial $X$.

## 1.2  Conventional method: classifying with CSP features

In the motor-imagery EEG signal classification, Common Spatial Pattern (CSP) based classifiers have proven to be powerful [11, 3, 6]. CSP is a decomposition method proposed by Koles [9] that finds a set of projections that simultaneously diagonalize the covariance matrices corresponding to two brain states. Formally, the covariance matrices[2] are defined as:

$$\Sigma_c = \frac{1}{|\mathcal{I}_c|} \sum_{i \in \mathcal{I}_c} X_i X_i^\top \qquad (c \in \{+, -\}), \tag{1}$$

where $\mathcal{I}_c$ is the set of indices belonging to a class $c \in \{+, -\}$; thus $\mathcal{I}_+ \cup \mathcal{I}_- = \{1, \ldots, n\}$. Then, the simultaneous diagonalization is achieved by solving the following generalized eigenvalue problem:

$$\Sigma_+ \boldsymbol{w} = \lambda \Sigma_- \boldsymbol{w}. \tag{2}$$

Note that for each pair of eigenvector and eigenvalue $(\boldsymbol{w}_j, \lambda_j)$, the equality $\lambda_j = \frac{\boldsymbol{w}_j^\top \Sigma_+ \boldsymbol{w}_j}{\boldsymbol{w}_j^\top \Sigma_- \boldsymbol{w}_j}$ holds. Therefore, the eigenvector with the largest eigenvalue corresponds to the projection with the maximum ratio of power for the "+" class and the "−" class, and the other-way-around for the eigenvector with the smallest eigenvalue. In this paper, we call these eigenvectors *filters*[3]; we call the eigenvector of an eigenvalue smaller (or larger) than one a *filter for the "+" class (or the "−" class)*, respectively, because the signal projected with them optimally (in the spirit of eigenvalues) captures the task related *de*-synchronization in each class. It is common practice that only the first $n_{\text{of}}$ largest eigenvectors and the last $n_{\text{of}}$ smallest eigenvectors are used to construct a low dimensional feature representation. The feature vector consists of logarithms of the projected signal powers and a Linear Discriminant Analysis (LDA) classifier is trained on the resulting feature vector. To summarize, the conventional CSP based classifier can be constructed as follows:

**How to build a CSP based classifier**:

1. Solve the generalized eigenvalue problem Eq. (2).
2. Take the $n_{\text{of}}$ largest and smallest eigenvectors $\{\boldsymbol{w}_j\}_{j=1}^J \quad (J = 2n_{\text{of}})$.
3. $\boldsymbol{x}_i := \left\{ \log \boldsymbol{w}_j^\top X_i X_i^\top \boldsymbol{w}_j \right\}_{j=1}^J \qquad (i = 1, \ldots, n)$.
4. Train an LDA classifier on $\{\boldsymbol{x}_i, y_i\}_{i=1}^n$.

## 2 Theory

### 2.1 The model

We consider the following *discriminative* model; we model the symmetric logit transform of the posterior class probability to be a linear function with respect to the second order statistics of the EEG signal:

$$\log \frac{P(y=+1|X)}{P(y=-1|X)} = f(X;\boldsymbol{\theta}) := \mathrm{tr}\left[WXX^\top\right] + b, \tag{3}$$

where $\boldsymbol{\theta} := (W, b) \in \mathrm{Sym}(d) \times \mathbb{R}$, $W$ is a symmetric $d \times d$ matrix and $b$ is the bias term.

The model (3) can be derived by assuming a zero-mean Gaussian distribution with no temporal correlation with a covariance matrix $\Sigma_\pm$ for each class as follows:

$$\log \frac{P(y=+1|X)}{P(y=-1|X)} = \frac{1}{2}\mathrm{tr}\left[\left(-\Sigma_+^{-1} + \Sigma_-^{-1}\right)XX^\top\right] + \mathrm{const..} \tag{4}$$

However training of a discriminative model is robust to misspecification of the marginal distribution $P(X)$ [13]. In another words, the marginal distribution $P(X)$ is a nuisance parameter; we maximize the joint log-likelihood, which is decomposed as $\log P(y, X|\boldsymbol{\theta}) = \log P(y|X, \boldsymbol{\theta}) + \log P(X)$, only with respect to $\boldsymbol{\theta}$ [14]. Therefore, no assumption about the generative model is necessary. Note that from Eq. (4) normally the optimal $W$ has both positive and negative eigenvalues.

### 2.2 Logistic regression

#### 2.2.1 Linear logistic regression

We minimize the negative log-likelihood of Eq. (3) with an additional regularization term, which is written as follows:

$$\min_{W \in \mathrm{Sym}(d), b \in \mathbb{R}} \frac{1}{n} \sum_{i=1}^{n} \log\left(1 + e^{-y_i f(X_i;\boldsymbol{\theta})}\right) + \frac{C}{2n}\left(\mathrm{tr}\Sigma_P W \Sigma_P W + b^2\right). \tag{5}$$

Here, the pooled covariance matrix $\Sigma_P := \frac{1}{n}\sum_{i=1}^{n} X_i X_i^\top$ is introduced in the regularization term in order to make the regularization invariant to linear transformation of the data; if we rewrite $W$ as $W := \Sigma_P^{-1/2}\tilde{W}\Sigma_P^{-1/2}$, one can easily see that the regularization term is simply the Frobenius norm of a symmetric matrix $\tilde{W}$; the transformation corresponds to the whitening of the signal $\tilde{X} = \Sigma_P^{-1/2}X$. By simple calculation, one can see that the loss term is the negative logarithm of the conditional likelihood $\prod_{i=1}^{n} 1/(1 + e^{-y_i f(X_i;\boldsymbol{\theta})})$, in another words the probability of observing head ($y_i = +1$) or tail ($y_i = -1$) by tossing $n$ coins with probability $P(y = +1|X = X_i, \boldsymbol{\theta})$ ($i = 1, \ldots, n$) for the head. From a general point of view, the loss term of Eq. (5) converges asymptotically to the true loss where the empirical average is replaced by the expectation over $X$ and $y$, whose minimum over functions in $L^2(P_X)$ is achieved by the symmetric logit transform of $P(y = +1|X)$ [15].

Note that the problem Eq. (5) is *convex*. The problem of classifying motor imagery EEG signals is now addressed under a single loss function. Based on the criterion (Eq. (5)) we can say how good a solution is and we know how to properly regularize it.

#### 2.2.2 Rank=2 approximation of the linear logistic regression

Here we present a rank=2 approximation of the regression function (3). Using this approximation we can greatly reduce the number of parameters to be estimated from a symmetric matrix coefficient to a pair of projection coefficients and additionally gain insight into the relevant feature the classifier has found.

The rank=2 approximation of the regression function (3) is written as follows:

$$\bar{f}(X;\bar{\boldsymbol{\theta}}) := \frac{1}{2}\mathrm{tr}\left[\left(-\boldsymbol{w}_1\boldsymbol{w}_1^\top + \boldsymbol{w}_2\boldsymbol{w}_2^\top\right)XX^\top\right] + b, \tag{6}$$

where $\bar{\boldsymbol{\theta}} := (\boldsymbol{w}_1, \boldsymbol{w}_2, b) \in \mathbb{R}^d \times \mathbb{R}^d \times \mathbb{R}$. The rationale for choosing this special form of function is that the Bayes optimal regression coefficients in Eq. (4) is the difference of two positive definite matrices; therefore two bases with opposite signs are at least necessary in capturing the nature of Eq. (4) (incorporating more bases goes beyond the scope of this contribution).

The rank=2 parameterized logistic regression can be obtained by minimizing the sum of the logistic regression loss and regularization terms similarly to Eq. (5):

$$\min_{\boldsymbol{w}_1, \boldsymbol{w}_2 \in \mathbb{R}^d, b \in \mathbb{R}} \frac{1}{n} \sum_{i=1}^{n} \log\left(1 + e^{-y_i \bar{f}(X_i; \bar{\boldsymbol{\theta}})}\right) + \frac{C}{2n}\left(\boldsymbol{w}_1^\top \Sigma_P \boldsymbol{w}_1 + \boldsymbol{w}_2^\top \Sigma_P \boldsymbol{w}_2 + b^2\right). \qquad (7)$$

Here, again the pooled covariance matrix $\Sigma_P$ is used as a metric in order to ensure the invariance to linear transformations. Note that the bases $\{\boldsymbol{w}_1, \boldsymbol{w}_2\}$ give projections of the signal into a two dimensional feature space in a similar manner as CSP (see Sec. 1.2). We call $\boldsymbol{w}_1$ and $\boldsymbol{w}_2$ filters corresponding to "+" and "−" classes, respectively, similarly to CSP. The filters can be topographically mapped onto the scalp, from which insight into the classifier can be obtained. However, the major difference between CSP and the rank=2 parameterized logistic regression (Eq. (7)) is that in our new approach, there is no distinction between the feature extraction step and the classifier training step. The coefficient that linearly combines the features (i.e., the norm of $\boldsymbol{w}_1$ and $\boldsymbol{w}_2$) is optimized in the same optimization problem (Eq. (7)).

## 3 Results

### 3.1 Experimental settings

We compare the logistic regression classifiers (Eqs. (3) and (6)) against CSP based classifiers with $n_{\mathrm{of}} = 1$ (total 2 filters) and $n_{\mathrm{of}} = 3$ (total 6 filters). The comparison is a chronological validation. All methods are trained on the first half of the samples and applied on the second half. We use 60 BCI experiments [6] from 29 subjects where the subjects performed three imaginary movements, namely "right hand" (R), "left hand" (L) and "foot" (F) according to the visual cue presented on the screen, except 9 experiments where only two classes were performed. Since we focus on binary classification, all the pairwise combination of the performed classes produced $162 (= 51 \cdot 3 + 9)$ datasets. Each dataset contains 70 to 600 trials (at median 280) of imaginary movements. All the recordings come from the calibration measurements, i.e. no feedback was presented to the subjects. The signal was recorded from the scalp with multi-channel EEG amplifiers using 32, 64 or 128 channels. The signal was sampled at 1000Hz and down-sampled to 100Hz before the processing.

The signal is band-pass filtered at 7-30Hz and the interval 500-3500ms after the appearance of visual cue is cut out from the continuous EEG signal as a trial $X$. The training data is whitened before minimizing Eqs. (5) and (7) because both problems become considerably simpler when $\Sigma_P$ is an identity matrix. For the prediction of test data, coefficients including the whitening operation $W = \Sigma_P^{-1/2} \tilde{W} \Sigma_P^{-1/2}$ for Eq. (3) and $\boldsymbol{w}_j = \Sigma_P^{-1/2} \tilde{\boldsymbol{w}}_j$ ($j = 1, 2$) for Eq. (6) are used, where $\tilde{W}$ and $\tilde{\boldsymbol{w}}_j$ denote the minimizer of Eqs. (5) and (7) for the whitened data. Note that we did *not* whitened the training and test data *jointly*, which could have improved the performance. The regularization constant $C$ for the proposed method is chosen by 5×10 cross-validation on the training set.

### 3.2 Classification performance

In Fig. 1, logistic regression (LR) classifiers with the full rank parameterization (Eq. (3); left column) and the rank=2 parameterization (Eq. (6); right column) are compared against CSP based classifiers with 6 filters (top row) and 2 filters (bottom row). Each plot shows the bit-rates achieved by CSP (horizontal) and LR (vertical) for each dataset as a circle. Here the bit-rate (per decision) is defined based on the classification test error $p_{\mathrm{err}}$ as the capacity of a binary symmetric channel with the same error probability:

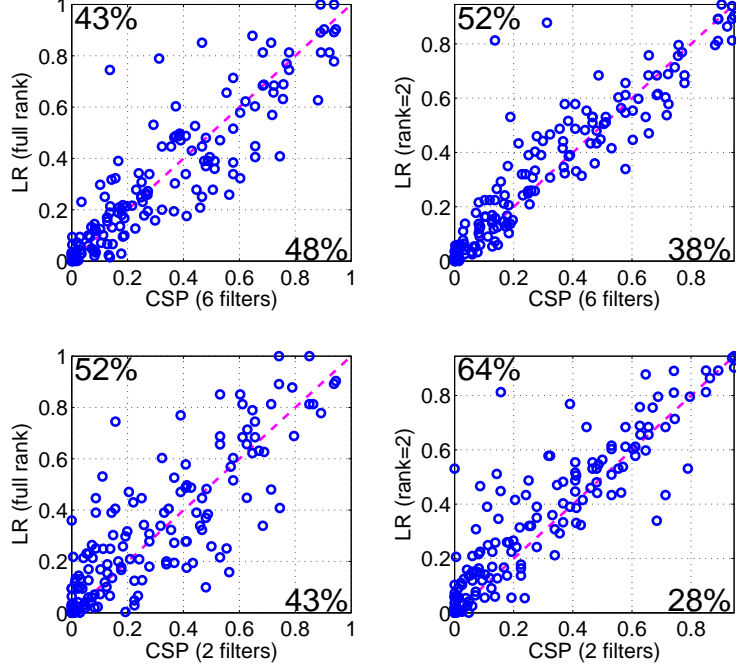

Figure 1: Comparison of bit-rates achieved by the CSP based classifiers and the logistic regression (LR) classifiers. The bit-rates achieved by the conventional CSP based classifier and the proposed LR classifier are shown as a circle for each dataset. The proportion of datasets lying above/below the diagonal is shown at top-left/bottom-right corners of each plot, respectively. Only the difference between CSP with 2 filters and rank=2 approximated LR (lower right) is significant based on Fisher sign test at 5% level.

$1 - \left( p_{\text{err}} \log_2 \frac{1}{p_{\text{err}}} + (1 - p_{\text{err}}) \log_2 \frac{1}{1 - p_{\text{err}}} \right)$. The proposed method improves upon the conventional method for datasets lying above the diagonal. Note that our proposed logistic regression ansatz is significantly better only in the lower right plot.

Figure 2 shows examples of spatial filter coefficients obtained by CSP (6 filters) and rank=2 parameterized logistic regression. The CSP filters for subject A (see Fig. 2(a)) include typical cases (the first filter for the "left hand" class and the first two filters for the "right hand" class) of filters corrupted by artifacts, e.g., muscle movements. The CSP filters for the "foot" class in subject B (see Fig. 2(b)) are corrupted by strong occipital $\alpha$-activity, which might have been weakly correlated to the labels by chance. Note that CSP with 2 filters only use the first filter for each class, which corresponds to the first row in Figs. 2(a) and 2(b). On the other hand the filter coefficients obtained by the logistic regression are clearly focused on the area physiologically corresponding to ERD in the motor cortex (see Figs. 2(c) and (d)).

## 4 Discussion

### 4.1 Relation to CSP

Here, we show that at the optimum of Eq. (7) the regression coefficients $\boldsymbol{w}_1$ and $\boldsymbol{w}_2$ are generalized eigenvectors of two *uncertainty weighted covariance matrices* corresponding to two motor imagery classes, which are weighted by the uncertainty of the decision $1 - P(y = y_i | X = X_i)$ for each sample. Samples that are easily explained by the regression function are weighed low whereas those lying close to the decision boundary or those lying on the wrong side of the boundary are highly weighted. Although, both CSP and the rank=2 approximated logistic regression can be understood as generalized eigenvalue decomposition, the classification-optimized weighting in the logistic regression yields filters that focus on

the task related modulation of rhythmic activities more clearly when compared to CSP, as shown in Fig. 2.

Differentiating Eq. (7) with either $\boldsymbol{w}_1$ or $\boldsymbol{w}_2$, we obtain the following equality which holds at the optimum.

$$\pm \sum_{i=1}^{n} \frac{e^{-z_i}}{1 + e^{-z_i}} y_i X_i X_i^{\top} \boldsymbol{w}_j^* + C \Sigma_P \boldsymbol{w}_j^* = 0 \qquad (j = 1, 2), \tag{8}$$

where we define the short hand $z_i := y_i \bar{f}(X_i; \bar{\boldsymbol{\theta}}^*)$ and $\pm$ denotes $+$ and $-$ for $j = 1, 2$, respectively. Moreover, Eq. (8) can be rewritten as follows:

$$\Sigma_-(\bar{\boldsymbol{\theta}}^*, 0)\boldsymbol{w}_1^* = \Sigma_+(\bar{\boldsymbol{\theta}}^*, C)\boldsymbol{w}_1^*, \tag{9}$$

$$\Sigma_+(\bar{\boldsymbol{\theta}}^*, 0)\boldsymbol{w}_2^* = \Sigma_-(\bar{\boldsymbol{\theta}}^*, C)\boldsymbol{w}_2^*, \tag{10}$$

where we define the *uncertainty weighted covariance matrix* as:

$$\Sigma_\pm(\bar{\boldsymbol{\theta}}^*, C) = \sum_{i \in \mathcal{I}_\pm} \frac{e^{-z_i}}{1 + e^{-z_i}} X_i X_i^{\top} + \frac{C}{n} \sum_{i=1}^{n} X_i X_i^{\top}.$$

Note that increasing the regularization constant $C$ biases the uncertainty weighted covariance matrix to the pooled covariance matrix $\Sigma_P$; the regularization only affects the right-hand side of Eqs. (9) and (10). If $C > 0$, the optimal filter coefficients $\boldsymbol{w}_j^*$ $(j = 1, 2)$ are the generalized eigenvectors of Eqs. (9) and (10), respectively.

## 4.2 CSP is not optimal

When first proposed, CSP was rather a decomposition technique than a classification technique (see [9]). After being introduced to the BCI community by [11], it has proved to be also powerful in classifying imaginary motor movements [3, 6]. However, since it is not optimized for the classification problem, there are two major drawbacks. Firstly, the selection of "good" CSP components is usually done somewhat arbitrarily. A widely used heuristic is to choose several generalized eigenvectors from both ends of the eigenvalue spectrum. However, as in subject B in Fig. 2, it is often observed that filters corresponding to overwhelming strong power come to the top of the spectrum though they are not correlated to the label so strongly. In practice, an experienced investigator can choose good filters by looking at them, however the validity of the selection cannot be assessed because the manual selection cannot be done inside the cross-validation. Secondly, simultaneous diagonalization of covariance matrices can suffer greatly from a few outlier trials as seen in subject A in Fig. 2. Again, in practice one can inspect the EEG signals to detect outliers, however a manual outlier detection is also a somewhat arbitrary, non-reproducible process, which cannot be validated.

## 5 Conclusion

In this paper, we have proposed an unified framework for single trial classification of motor-imagery EEG signals. The problem is addressed as a single minimization problem without any prior feature extraction or outlier removal steps. The task is to minimize a logistic regression loss with a regularization term. The regression function is a linear function with respect to the second order statistics of the EEG signal.

We have tested the proposed method on 162 BCI datasets. By parameterizing the whole regression coefficients directly, we have obtained comparable classification accuracy with CSP based classifiers. By parameterizing the regression coefficients as the difference of two rank-one matrices, improvement against CSP based classifiers was obtained.

We have shown that in the rank=2 parameterization of the logistic regression function, the optimal filter coefficients has an interpretation as a solution to a generalized eigenvalue problem similarly to CSP. However, the difference is that in the case of logistic regression every sample is weighted according to the importance to the overall classification problem whereas in CSP all the samples have uniform importance.

The proposed framework provides a basis for various future directions. For example, incorporating more than two filters will connect the two parameterizations of the regression function shown in this paper and it may allow us to investigate how many filters are sufficient for good classification. Since the classifier output is the logit transform of the class probability, it is straightforward to generalize the method to multi-class problems. Also non-stationarities, e.g. caused by a covariate shift (see [16, 17]) in the density $P(X)$ from one session to another, could be corrected by adapting the likelihood model.

**Acknowledgments**: This research was partially supported by MEXT, Grant-in-Aid for JSPS fellows, 17-11866 and Grant-in-Aid for Scientific Research on Priority Areas, 17022012, by BMBF-grant FKZ 01IBE01A, and by the IST Programme of the European Community, under the PASCAL Network of Excellence, IST-2002-506778. This publication only reflects the authors' views.

## Footnotes

[1]For simplicity, we assume that the signal is already band-pass filtered and each trial is centered and scaled as $X = \frac{1}{\sqrt{T}} X_{\text{original}} \left( I_T - \frac{1}{T} \mathbf{1}\mathbf{1}^\top \right)$.

[2]Although it is convenient to call Eq. (1) a *covariance matrix*, calling it an *averaged cross power matrix* gives better insight into the nature of the problem, because we are focusing on the task related modulation of rhythmic activities.

[3]according to the convention by [12].

# References

[1] J. R. Wolpaw, N. Birbaumer, D. J. McFarland, G. Pfurtscheller, and T. M. Vaughan, "Brain-computer interfaces for communication and control", *Clin. Neurophysiol.*, 113: 767–791, 2002.

[2] N. Birbaumer, N. Ghanayim, T. Hinterberger, I. Iversen, B. Kotchoubey, A. Kübler, J. Perelmouter, E. Taub, and H. Flor, "A spelling device for the paralysed", *Nature*, 398: 297–298, 1999.

[3] G. Pfurtscheller, C. Neuper, C. Guger, W. Harkam, R. Ramoser, A. Schlögl, B. Obermaier, and M. Pregenzer, "Current Trends in Graz Brain-computer Interface (BCI)", *IEEE Trans. Rehab. Eng.*, 8(2): 216–219, 2000.

[4] B. Blankertz, G. Curio, and K.-R. Müller, "Classifying Single Trial EEG: Towards Brain Computer Interfacing", in: T. G. Diettrich, S. Becker, and Z. Ghahramani, eds., *Advances in Neural Inf. Proc. Systems (NIPS 01)*, vol. 14, 157–164, 2002.

[5] B. Blankertz, G. Dornhege, C. Schäfer, R. Krepki, J. Kohlmorgen, K.-R. Müller, V. Kunzmann, F. Losch, and G. Curio, "Boosting Bit Rates and Error Detection for the Classification of Fast-Paced Motor Commands Based on Single-Trial EEG Analysis", *IEEE Trans. Neural Sys. Rehab. Eng.*, 11(2): 127–131, 2003.

[6] B. Blankertz, G. Dornhege, M. Krauledat, K.-R. Müller, V. Kunzmann, F. Losch, and G. Curio, "The Berlin Brain-Computer Interface: EEG-based communication without subject training", *IEEE Trans. Neural Sys. Rehab. Eng.*, 14(2): 147–152, 2006.

[7] G. Dornhege, J. del R. Millán, T. Hinterberger, D. McFarland, and K.-R. Müller, eds., *Towards Brain-Computer Interfacing*, MIT Press, 2006, in press.

[8] T. N. Lal, M. Schröder, T. Hinterberger, J. Weston, M. Bogdan, N. Birbaumer, and B. Schölkopf, "Support Vector Channel Selection in BCI", *IEEE Transactions Biomedical Engineering*, 51(6): 1003–1010, 2004.

[9] Z. J. Koles, "The quantitative extraction and topographic mapping of the abnormal components in the clinical EEG", *Electroencephalogr. Clin. Neurophysiol.*, 79: 440–447, 1991.

[10] G. Pfurtscheller and F. H. L. da Silva, "Event-related EEG/MEG synchronization and desynchronization: basic principles", *Clin. Neurophysiol.*, 110(11): 1842–1857, 1999.

[11] H. Ramoser, J. Müller-Gerking, and G. Pfurtscheller, "Optimal spatial filtering of single trial EEG during imagined hand movement", *IEEE Trans. Rehab. Eng.*, 8(4): 441–446, 2000.

[12] N. J. Hill, J. Farquhar, T. N. Lal, and B. Schölkopf, "Time-dependent demixing of task-relevant EEG sources", in: *Proceedings of the 3rd International Brain-Computer Interface Workshop and Training Course 2006*, Verlag der Technischen Universität Graz, 2006.

[13] B. Efron, "The Efficiency of Logistic Regression Compared to Normal Discriminant Analysis", *J. Am. Stat. Assoc.*, 70(352): 892–898, 1975.

[14] T. Minka, "Discriminative models, not discriminative training", *Tech. Rep. TR-2005-144*, Microsoft Research Cambridge, 2005.

[15] T. Hastie, R. Tibshirani, and J. Friedman, *The Elements of Statistical Learning*, Springer-Verlag, 2001.

[16] H. Shimodaira, "Improving predictive inference under covariate shift by weighting the log-likelihood function", *Journal of Statistical Planning and Inference*, 90: 227–244, 2000.

[17] S. Sugiyama and K.-R. Müller, "Input-Dependent Estimation of Generalization Error under Covariate Shift", *Statistics and Decisions*, 23(4): 249–279, 2005.

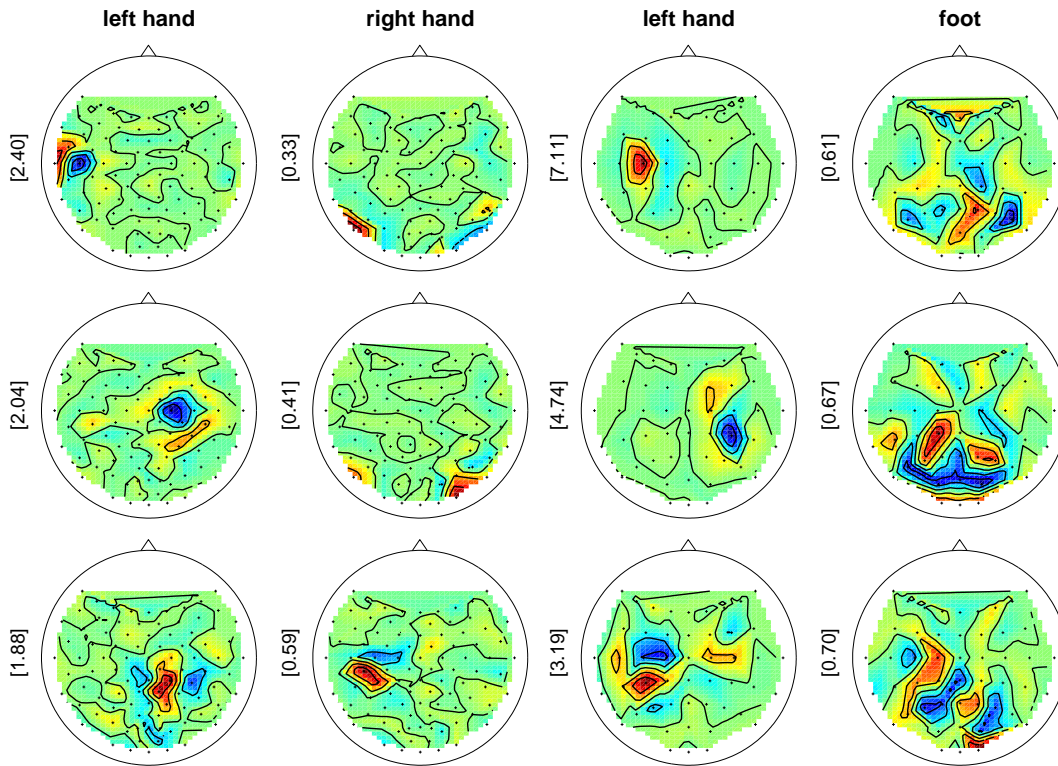

(a) Subject A. CSP filter coefficients

(b) Subject B. CSP filter coefficients

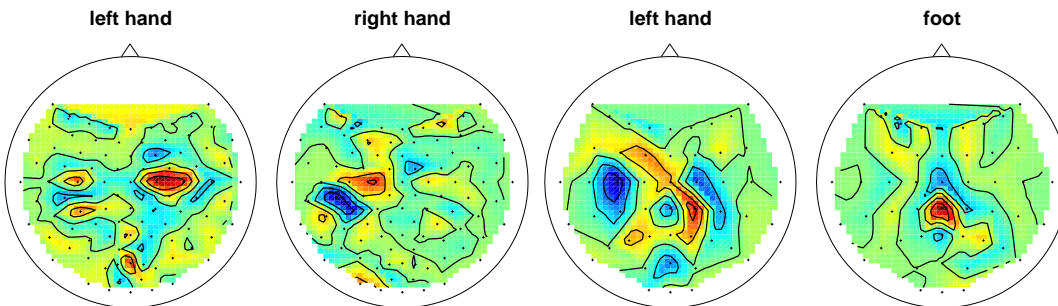

(c) Subject A. Logistic regression (rank=2)
filter coefficients

(d) Subject B. Logistic regression (rank=2)
filter coefficients

Figure 2: Examples of spatial filter coefficients obtained by CSP and the rank=2 parameterized logistic regression. (a) Subject A. Some CSP filters are corrupted by artifacts. (b) Subject B. Some CSP filters are corrupted by strong occipital $\alpha$-activity. (c) Subject A. Logistic regression coefficients are focusing on the physiologically expected "left hand" and "right hand" areas. (d) Subject B. Logistic regression coefficients are focusing on the "left hand" and "foot" areas. Electrode positions are marked with crosses in every plot. For CSP filters, the generalized eigenvalues (Eq. (2)) are shown inside brackets.